# An Efficient Sequential Monte Carlo Algorithm for Coalescent Clustering

**Dilan Görür**
Gatsby Unit
University College London
dilan@gatsby.ucl.ac.uk

**Yee Whye Teh**
Gatsby Unit
University College London
ywteh@gatsby.ucl.ac.uk

## Abstract

We propose an efficient sequential Monte Carlo inference scheme for the recently proposed coalescent clustering model [1]. Our algorithm has a quadratic runtime while those in [1] is cubic. In experiments, we were surprised to find that in addition to being more efficient, it is also a better sequential Monte Carlo sampler than the best in [1], when measured in terms of variance of estimated likelihood and effective sample size.

## 1 Introduction

Algorithms for automatically discovering hierarchical structure from data play an important role in machine learning. In many cases the data itself has an underlying hierarchical structure whose discovery is of interest, examples include phylogenies in biology, object taxonomies in vision or cognition, and parse trees in linguistics. In other cases, even when the data is not hierarchically structured, such structures are still useful simply as a statistical tool to efficiently pool information across the data at different scales; this is the starting point of hierarchical modelling in statistics.

Many hierarchical clustering algorithms have been proposed in the past for discovering hierarchies. In this paper we are interested in a Bayesian approach to hierarchical clustering [2, 3, 1]. This is mainly due to the appeal of the Bayesian approach being able to capture uncertainty in learned structures in a coherent manner. Unfortunately, inference in Bayesian models of hierarchical clustering are often very complex to implement, and computationally expensive as well.

In this paper we build upon the work of [1] who proposed a Bayesian hierarchical clustering model based on Kingman's coalescent [4, 5]. [1] proposed both greedy and sequential Monte Carlo (SMC) based agglomerative clustering algorithms for inferring hierarchical clustering which are simpler to implement than Markov chain Monte Carlo methods. The algorithms work by starting with each data item in its own cluster, and iteratively merge pairs of clusters until all clusters have been merged. The SMC based algorithm has computational cost $O(n^3)$ per particle, where $n$ is the number of data items.

We propose a new SMC based algorithm for inference in the coalescent clustering of [1]. The algorithm is based upon a different perspective on Kingman's coalescent than that in [1], where the computations required to consider whether to merge each pair of clusters at each iteration is not discarded in subsequent iterations. This improves the computational cost to $O(n^2)$ per particle, allowing this algorithm to be applied to larger datasets. In experiments we show that our new algorithm achieves improved costs without sacrificing accuracy or reliability.

Kingman's coalescent originated in the population genetics literature, and there has been significant interest there on inference, including Markov chain Monte Carlo based approaches [6] and SMC approaches [7, 8]. The SMC approaches have interesting relationship to our algorithm and to that of [1]. While ours and [1] integrate out the mutations on the coalescent tree and sample the coalescent

times, [7, 8] integrate out the coalescent times, and sample mutations instead. Because of this difference, ours and that of [1] will be more efficient in higher dimensional data, as well as other cases where the state space is too large and sampling mutations will be inefficient.

In the next section, we review Kingman's coalescent and the existing SMC algorithms for inference on this model. In Section 3, we describe a cheaper SMC algorithm. We compare our method with that of [1] in Section 4 and conclude with a discussion in Section 5.

## 2 Hierarchical Clustering using Kingman's Coalescent

Kingman's coalescent [4, 5] describes the family relationship between a set of haploid individuals by constructing the genealogy backwards in time. Ancestral lines coalesce when the individuals share a common ancestor, and the genealogy is a binary tree rooted at the common ancestor of all the individuals under consideration. We briefly review the coalescent and the associated clustering model as presented in [1] before presenting a different formulation more suitable for our proposed algorithm.

Let $\pi$ be the genealogy of $n$ individuals. There are $n-1$ coalescent events in $\pi$, we order these events with $i = 1$ being the most recent one, and $i = n - 1$ for the last event when all ancestral lines are coalesced. Event $i$ occurs at time $T_i < 0$ in the past, and involves the coalescing of two ancestors, denoted $\rho_{l_i}$ and $\rho_{r_i}$, into one denoted $\rho_i$. Let $A_i$ be the set of ancestors right after coalescent event $i$, and $A_0$ be the full set of individuals at the present time $T_0 = 0$. To draw a sample $\pi$ from Kingman's coalescent we sample the coalescent events one at a time starting from the present. At iteration $i$ we pick the pair of individuals $\rho_{l_i}$, $\rho_{r_i}$ uniformly at random from the $n - i + 1$ individuals available in $A_{i-1}$, pick a waiting time $\delta_i \sim \text{Exp}(\binom{n-i+1}{2})$ from an exponential distribution with rate $\binom{n-i+1}{2}$ equal to the number of pairs available, and set $A_i = A_{i-1} - \{\rho_{l_i}, \rho_{r_i}\} + \{\rho_i\}$, $T_i = T_{i-1} - \delta_i$. The probability of $\pi$ is thus:

$$p(\pi) \quad = \quad \prod_{i=1}^{n-1} \exp\left( -\binom{n-i+1}{2}\delta_i \right). \tag{1}$$

The coalescent can be used as a prior over binary trees in a model where we have a tree-structured likelihood function for observations at the leaves. Let $\theta_i$ be the subtree rooted at $\rho_i$ and $\mathbf{x}_i$ be the observations at the leaves of $\theta_i$. [1] showed that by propagating messages up the tree the likelihood function can be written in a sequential form:

$$p(\mathbf{x} \,|\, \pi) = Z_0(\mathbf{x}) \prod_{i=1}^{n-1} Z_{\rho_i}(\mathbf{x}_i|\theta_i), \tag{2}$$

where $Z_{\rho_i}$ is a function only of the coalescent times associates with $\rho_{l_i}$, $\rho_{r_i}$, $\rho_i$ and of the local messages sent from $\rho_{l_i}$, $\rho_{r_i}$ to $\rho_i$, and $Z_0(\mathbf{x})$ is an easily computed normalization constant in eq. (2). Each function has the form (see [1] for further details):

$$Z_{\rho_i}(\mathbf{x}_i|\theta_i) = \int p_0(y_i) \prod_{c=l_i,r_i} \int p(y_c|y_i, \theta_i) M_{\rho_c}(y_c) \, dy_c \, dy_i \tag{3}$$

where $M_{\rho_c}$ is the message from child $\rho_c$ to $\rho_i$. The posterior is proportional to the product of eq. (1) and eq. (2) and our aim is to have an efficient way to compute the posterior. For this purpose, we will give a different perspective to constructing the coalescent in the following and describe our sequential Monte Carlo algorithm in Section 3.

### 2.1 A regenerative race process

In this section we describe a different formulation of the coalescent based on the fact that each stage of the coalescent can be interpreted as a race between the $\binom{n-i+1}{2}$ pairs of individuals to coalesce. Each pair proposes a coalescent time, the pair with most recent coalescent time "wins" the race and gets to coalesce, at which point the next stage starts with $\binom{n-i}{2}$ pairs in the race. Naïvely this race process would require a total of $O(n^3)$ pairs to propose coalescent times. We show that using the regenerative (memoryless) property of exponential distributions allows us to reduce this to $O(n^2)$.

---

**Algorithm 1** A regenerative race process for constructing the coalescent

    **inputs:** number of individuals $n$,
    set starting time $T_0 = 0$ and $A_0$ the set of $n$ individuals
    **for all** pairs of existing individuals $\rho_l, \rho_r \in A_0$ **do**
        propose coalescent time $t_{lr}$ using eq. (4)
    **end for**
    **for all** coalescence events $i = 1 : n - 1$ **do**
        find the pair to coalesce $(\rho_{l_i}, \rho_{r_i})$ using eq. (5)
        set coalescent time $T_i = t_{l_i r_i}$ and update $A_i = A_{i-1} - \{\rho_{l_i}, \rho_{r_i}\} + \{\rho_i\}$
        remove pairs with $\rho_l \in \{\rho_{l_i}, \rho_{r_i}\}, \rho_r \in A_{i-1} \backslash \{\rho_{l_i}, \rho_{r_i}\}$
        **for all** new pairs with $\rho_l = \rho_i, \rho_r \in A_i \backslash \{\rho_i\}$ **do**
            propose coalescent time using eq. (4)
        **end for**
    **end for**

---

The same idea will allow us to reduce the computational cost of our SMC algorithm from $O(n^3)$ to $O(n^2)$.

At stage $i$ of the coalescent we have $n - i + 1$ individuals in $A_{i-1}$, and $\binom{n-i+1}{2}$ pairs in the race to coalesce. Each pair $\rho_l, \rho_r \in A_{i-1}, \rho_l \neq \rho_r$ proposes a coalescent time

$$t_{lr}|T_{i-1} \sim T_{i-1} - \text{Exp}(1), \tag{4}$$

that is, by subtracting from the last coalescent time a waiting time drawn from an exponential distribution of rate 1. The pair $\rho_{l_i}, \rho_{r_i}$ with most recent coalescent time wins the race:

$$(\rho_{l_i}, \rho_{r_i}) = \underset{(\rho_l, \rho_r)}{\text{argmax}} \left\{ t_{lr}, \quad \rho_l, \rho_r \in A_{i-1}, \rho_l \neq \rho_r \right\} \tag{5}$$

and coalesces into a new individual $\rho_i$ at time $T_i = t_{l_i r_i}$. At this point stage $i + 1$ of the race begins, with some pairs dropping out of the race (specifically those with one half of the pair being either $\rho_{l_i}$ or $\rho_{r_i}$) and new ones entering (specifically those formed by pairing the new individual $\rho_i$ with an existing one). Among the pairs $(\rho_l, \rho_r)$ that did not drop out nor just entered the race, consider the distribution of $t_{lr}$ conditioned on the fact that $t_{lr} < T_i$ (since $(\rho_l, \rho_r)$ did not win the race at stage $i$). Using the memoryless property of the exponential distribution, we see that $t_{lr}|T_i \sim T_i - \text{Exp}(1)$, thus eq. (4) still holds and *we need not redraw $t_{lr}$ for the stage $i + 1$ race.* In other words, once $t_{lr}$ is drawn once, it can be reused for subsequent stages of the race until it either wins a race or drops out. The generative process is summarized in Algorithm 1.

We obtain the probability of the coalescent $\pi$ as a product over the $i = 1, \ldots, n - 1$ stages of the race, of the probability of each event "$\rho_{l_i}, \rho_{r_i}$ wins stage $i$ and coalesces at time $T_i$" given more recent stages. The probability at stage $i$ is simply the probability that $t_{l_i r_i} = T_i$, and that all other proposed coalescent times $t_{lr} < T_i$, conditioned on the fact that the proposed coalescent times $t_{lr}$ for all pairs at stage $i$ are all less than $T_{i-1}$. This gives:

$$p(\pi) = \prod_{i=1}^{n-1} \left( p(t_{l_i r_i} = T_i \,|\, t_{l_i r_i} < T_{i-1}) \prod_{(\rho_l, \rho_r) \neq (\rho_{l_i}, \rho_{r_i})} p(t_{l'r'} < T_i \,|\, t_{l'r'} < T_{i-1}) \right) \tag{6}$$

$$= \prod_{i=1}^{n-1} \left( \frac{p(t_{l_i r_i} = T_i)}{p(t_{l_i r_i} < T_{i-1})} \prod_{(\rho_l, \rho_r) \neq (\rho_{l_i}, \rho_{r_i})} \frac{p(t_{lr} < T_i)}{p(t_{lr} < T_{i-1})} \right) \tag{7}$$

where the second product runs over all pairs in stage $i$ except the winning pair. Each pair that participated in the race has corresponding terms in eq. (7), starting at the stage when the pair entered the race, and ending with the stage when the pair either dropped out or wins the stage. As these terms cancel, eq. (7) simplifies to,

$$p(\pi) = \prod_{i=1}^{n-1} \left( p(t_{l_i r_i} = T_i) \prod_{\rho_l \in \{\rho_{l_i}, \rho_{r_i}\}, \rho_r \in A_{i-1} \backslash \{\rho_{l_i}, \rho_{r_i}\}} p(t_{lr} < T_i) \right), \tag{8}$$

where the second product runs only over those pairs that dropped out after stage $i$. The first term is the probability of pair $(\rho_{l_i}, \rho_{r_i})$ coalescing at time $T_i$ given its entrance time, and the second term is the probability of pair $(\rho_l, \rho_r)$ dropping out of the race at time $T_i$ given its entrance time. We can verify that this expression equals eq. (1) by plugging in the probabilities for exponential distributions. Finally, multiplying the prior eq. (8) and the likelihood eq. (2) we have,

$$p(\mathbf{x}, \pi) = Z_0(\mathbf{x}) \prod_{i=1}^{n-1} \left( Z_{\rho_i}(\mathbf{x}_i | \theta_i) p(t_{l_i r_i} = T_i) \prod_{\rho_l \in \{\rho_{l_i}, \rho_{r_i}\}, \, \rho_r \in A_{i-1} \setminus \{\rho_{l_i}, \rho_{r_i}\}} p(t_{lr} < T_i) \right). \quad (9)$$

## 3 Efficient SMC Inference on the Coalescent

Our sequential Monte Carlo algorithm for posterior inference is directly inspired by the regenerative race process described above. In fact the algorithm is structurally exactly as in Algorithm 1, but with each pair $\rho_l$, $\rho_r$ proposing a coalescent time from a proposal distribution $t_{lr} \sim Q_{lr}$ instead of from eq. (4). The idea is that the proposal distribution $Q_{lr}$ is constructed taking into account the observed data, so that Algorithm 1 produces better approximate samples from the posterior.

The overall probability of proposing $\pi$ under the SMC algorithm can be computed similarly to eq. (6)-(8), and is,

$$q(\pi) = \prod_{i=1}^{n-1} \left( q_{l_i r_i}(t_{l_i r_i} = T_i) \prod_{\rho_l \in \{\rho_{l_i}, \rho_{r_i}\}, \rho_r \in A_{i-1} \setminus \{\rho_{l_i}, \rho_{r_i}\}} q_{lr}(t_{lr} < T_i) \right), \quad (10)$$

where $q_{lr}$ is the density of $Q_{lr}$. As both eq. (9) and eq. (10) can be computed sequentially, the weight $w$ associated with each sample $\pi$ can be computed "on the fly" as the coalescent tree is constructed:

$$w_0 = Z_0(\mathbf{x})$$
$$w_i = w_{i-1} \frac{Z_{\rho_i}(\mathbf{x}_i | \theta_i) p(t_{l_i r_i} = T_i)}{q_{l_i r_i}(t_{l_i r_i} = T_i)} \prod_{\rho_l \in \{\rho_{l_i}, \rho_{r_i}\}, \, \rho_r \in A_{i-1} \setminus \{\rho_{l_i}, \rho_{r_i}\}} \frac{p(t_{lr} < T_i)}{q_{lr}(t_{lr} < T_i)}. \quad (11)$$

Finally we address the choice of proposal distribution $Q_{lr}$ to use. [1] noted that $Z_{\rho_i}(\mathbf{x}_i | \theta_i)$ acts as a "local likelihood" term in eq. (9). We make use of this observation and use eq. (4) as a "local prior", i.e. the following density for the proposal distribution $Q_{lr}$:

$$q_{lr}(t_{lr}) \propto Z_{\rho_{lr}}(\mathbf{x}_{lr} | t_{lr}, \rho_l, \rho_r, \theta_{i-1}) p(t_{lr} \, | \, T_{c(lr)}) \quad (12)$$

where $\rho_{lr}$ is a hypothetical individual resulting from coalescing $l$ and $r$, $T_{c(lr)}$ denotes the time when the pair $(\rho_l, \rho_r)$ enters the race, $\mathbf{x}_{lr}$ are the data under $\rho_l$ and $\rho_r$, and $p(t_{lr} \, | \, T_{c(lr)}) = e^{t_{lr} - T_{c(lr)}} \mathbb{I}(t_{lr} < T_{c(lr)})$ is simply an exponential density with rate 1 that has been shifted and reflected. $\mathbb{I}(\cdot)$ is an indicator function returning 1 if its argument is true, and 0 otherwise.

The proposal distribution in [1] also has a form similar to eq. (12), but with the exponential rate being $\binom{n-i+1}{2}$ instead, if the proposal was in stage $i$ of the race. This dependence means that at each stage of the race the coalescent times proposal distribution needs to be recomputed for each pair, leading to an $O(n^3)$ computation time. On the other hand, similar to the prior process, we need to propose a coalescent time for each pair only once when it is first created. This results in $O(n^2)$ computational complexity per particle[1].

Note that it may not always be possible (or efficient) to compute the normalizing constant of the density in eq. (12) (even if we can sample from it efficiently). This means that the weight updates eq. (11) cannot be computed. In that case, we can use an approximation $\tilde{Z}_{\rho_{lr}}$ to $Z_{\rho_{lr}}$ instead. In the following subsection we describe the independent-sites parent-independent model we used in the experiments, and how to construct $\tilde{Z}_{\rho_{lr}}$.

### 3.1 Independent-Sites Parent-Independent Likelihood Model

In our experiments we have only considered coalescent clustering of discrete data, though our approach can be applied more generally. Say each data item consists of a $D$ dimensional vector where each entry can take on one of $K$ values. We use the independent-sites parent-independent mutation model over multinomial vectors in [1] as our likelihood model. Specifically, this model assumes that each point on the tree is associated with a $D$ dimensional multinomial vector, and each entry of this vector on each branch of the tree evolves independently (thus independent-sites), forward in time, and with mutations occurring at rate $\lambda_d$ on entry $d$. When a mutation occurs, a new value for the entry is drawn from a distribution $\phi_d$, independently of the previous value at that entry (thus parent-independent). When a coalescent event is encountered, the mutation process evolves independently down both branches.

Some calculations show that the transition probability matrix of the mutation process associated with entry $d$ on a branch of length $t$ is $e^{-\lambda_d t} I_K + (1 - e^{-\lambda_d t}) \phi_d^\top \mathbf{1}_K$, where $I_K$ is the identity matrix, $\mathbf{1}_K$ is a vector of 1's, and we have implicitly represented the multinomial distribution $\phi_d$ as a vector of probabilities. The message for entry $d$ from node $\rho_i$ on the tree to its parent is a vector $M_{\rho_i}^d = [M_{\rho_i}^{d1}, \ldots, M_{\rho_i}^{dK}]^\top$, normalized so that $\phi_d^\top M_{\rho_i}^d = 1$. The local likelihood term is then:

$$Z_{\rho_{lr}}^d(\mathbf{x}_{lr}|t_{lr}, \rho_l, \rho_r, \theta_{i-1}) = 1 - e^{\lambda_d(2t_{lr} - t_l - t_r)}\left(1 - \sum_{k=1}^K \phi_{dk} M_{\rho_l}^{dk} M_{\rho_r}^{dk}\right) \tag{13}$$

The logarithm of the proposal density is then:

$$\log q_{lr}(t_{lr}) = \text{constant} + (t_{lr} - T_{c(lr)}) + \sum_{d=1}^D \log Z_{\rho_{lr}}^d(\mathbf{x}_{lr}|t_{lr}, \rho_l, \rho_r, \theta_{i-1}) \tag{14}$$

This is not of standard form, and we use an approximation $\log \tilde{q}_{lr}(t_{lr})$ instead. Specifically, we use a piecewise linear $\log \tilde{q}_{lr}(t_{lr})$, which can be easily sampled from, and for which the normalization term is easy to compute.

The approximation is constructed as follows. Note that $\log Z_{\rho_{lr}}^d(\mathbf{x}_{lr}|t_{lr}, \rho_l, \rho_r, \theta_{i-1})$, as a function of $t_{lr}$, is concave if the term inside the parentheses in eq. (13) is positive, convex if negative, and constant if zero. Thus eq. (14) is a sum of linear, concave and convex terms. Using the upper and lower envelopes developed for adaptive rejection sampling [9], we can construct piecewise linear upper and lower envelopes for $\log q_{lr}(t_{lr})$ by upper and lower bounding the concave and convex parts separately. The upper and lower envelopes give exact bounds on the approximation error introduced, and we can efficiently improve the envelopes until a given desired approximation error is achieved. Finally, we used the upper bound as our approximate $\log \tilde{q}_{lr}(t_{lr})$. Note that the same issue arises in the proposal distribution for `SMC-PostPost`, and we used the same piecewise linear approximation. The details of this algorithm can be found in [10].

## 4 Experiments

The improved computational cost of inference makes it possible to do Bayesian inference for the coalescence models on larger datasets. The SMC samplers converge to the exact solution in the limit of infinite particles. However, it is not enough to be more efficient per particle, the crucial point is how efficient the algorithm is overall. An important question is how many particles we need in practice. To address this question, we compared the performance of our algorithm `SMC1` to `SMC-PostPost` on the synthetic data shown in Figure 1[2]. There are 15 binary 12-dimensional vectors in the dataset. There is overlap between the features of the data points however the data does not obey a tree structure, which will result in a multimodal posterior. Both `SMC1` and `SMC-PostPost` recover the structure with only a few particles. However there is room for improvement as the variance in the likelihood obtained from multiple runs decreases with increasing number of particles. Since both SMC algorithms are exact in the limit, the values should converge as we add more particles. We can check convergence by observing the variance of likelihood estimates of multiple runs. The variance

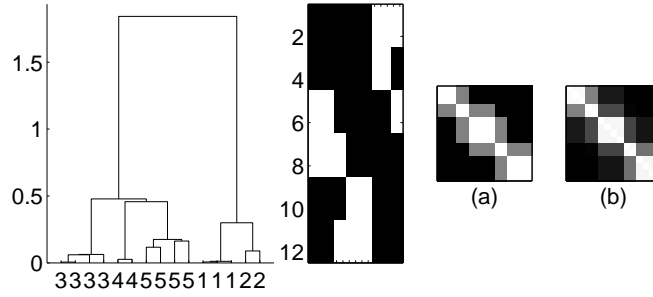

Figure 1: Synthetic data features is shown on the left; each data point is a binary column vector. A sample tree from the `SMC1` algorithm demonstrate that the algorithm could capture the similarity structure. The true covariance of the data (a) and the distance on the tree learned by the `SMC1` algorithm averaged over particles (b) are shown, showing that the overall structure was corerctly captured. The results obtained from `SMC-PostPost` were very similar to `SMC1` therefore are not shown here.

should shrink as we increase the number of particles. Figure 2 shows the change in the estimated likelihood as a function of number of particles. From this figure, we can conclude that the computationally cheaper algorithm `SMC1` is more efficient also in the number of particles as it gives more accurate answers with less particles.

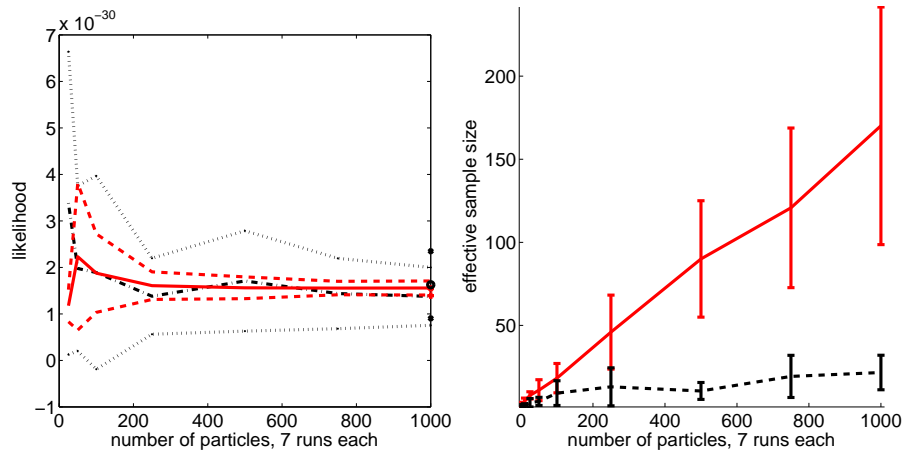

Figure 2: The change in the likelihood (left) and the effective sample size (right) as a function of number of particles for `SMC1` (solid) and `SMC-PostPost` (dashed). The mean estimate of both algorithms are very close, with the `SMC1` having a much tighter variance. The variance of both algorithms shrink and the effective sample size increases as the number of particles increase.

A quantity of interest in genealogical studies is the time to the most recent common ancestor (MRCA), which is the time of the last coalescence event. Although there is not a physical interpretation of this quantity for hierarchical clustering, it gives us an indication about the variance of the particles. We can observe the variation in the time to MRCA to assess convergence. Similar to the variance behaviour in the likelihood, with small number of particles `SMC-PostPost` has higher variance than `SMC1` . However, as there are more particles, results of the two algorithms almost overlap. The mean time for each step of coalescence together with its variance for 7250 particles for both algorithms is depicted in Figure 3. It is interesting that the first few coalescence times of `SMC1` are shorter than those for `SMC-PostPost`. The distribution of the particle weights is important for the efficiency of the importance sampler. Ideally, the weights would be uniform such that each particle contributes equally to the posterior estimation. If there is only a few particles that come from a high probability region, the weights of those particles would be much larger than

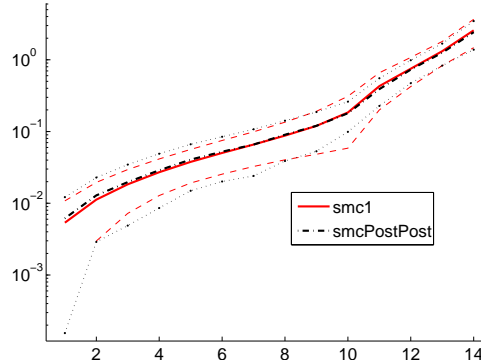

Figure 3: Times for each coalescence step averaged over 7250 particles. Note that both algorithms almost converged at the same distribution when given enough resources. There is a slight difference in the mean coalescence time. It is interesting that the `SMC1` algorithm proposes shorter times for the initial coalescence events.

the rest, resulting in a low effective sample size. We will discuss this point more in the next section. Here, we note that for the synthetic dataset, the effective sample size of `SMC-PostPost` is very poor, and that of `SMC1` is much higher, see Figure 2.

## 5  Discussion

We described an efficient Sequential Monte Carlo algorithm for inference on hierarchical clustering models that use Kingman's coalescent as a proir. Our method makes use of a regenerative perspective to construct the coalescent tree. Using this construction, we achieve quadratic run time per particle. By employing a tight upper bound on the local likelihood term, the proposed algorithm is applicable to general data generation processes.

We also applied our algorithm for inferring the structure in the phylolinguistic data used in [1]. We used the same Indo-European subset of the data, with the same subset of features, that is 44 languages with 100 binary features. Three example trees with the largest weights out of 7750 samples are depicted in Figure 4. Unfortunately, on this dataset, the effective sample size of both algorithms is close to one. A usual method to circumvent the low effective sample size problem in sequential Monte Carlo algorithms is to do resampling, that is, detecting the particles that will not contribute much to the posterior from the partial samples and prune them away, multiplying the promising samples. There are two stages to doing resampling. We need to decide at what point to prune away samples, and how to select which samples to prune away. As shown by [11], different problems may require different resampling algorithms. We tried resampling using Algorithm 5.1 of [12], however this only had a small improvement in the final performance for both algorithms on this data set.

Note that both algorithms use "local likelihoods" for calculating the weights, therefore the weights are not fully informative about the actual likelihood of the partial sample. Furthermore, in the recursive calculation of the weights in `SMC1` , we are including the effect of a pair only when they either coalesce or cease to exist for the sake of saving computations. Therefore the partial weights are even less informative about the state of the sample and the effective sample size cannot really give full explanation about whether the current sample is good or not. In fact, we did observe oscillations on the effective sample size calculated on the weights along the iterations, i.e. starting off with a high value, decreasing to virtually 1 and increasing later before the termination, which also indicates that it is not clear which of the particles will be more effective eventually. An open question is how to incorporate a resampling algorithm to improve the efficiency.

## Footnotes

[1]Technically the time cost is $O(n^2(m + \log n))$, where $n$ is the number of individuals, and $m$ is the cost of sampling from and evaluating eq. (12). The additional $\log n$ factor comes about because a priority queue needs to be maintained to determine the winner of each stage efficiently, but this is negligible compared to $m$.

[2]The comparison is done in the importance sampling setting, i.e. without using resampling for comparison of the proposal distributions.

## References

[1] Y. W. Teh, H. Daume III, and D. M. Roy. Bayesian agglomerative clustering with coalescents. In *Advances in Neural Information Processing Systems*, volume 20, 2008.

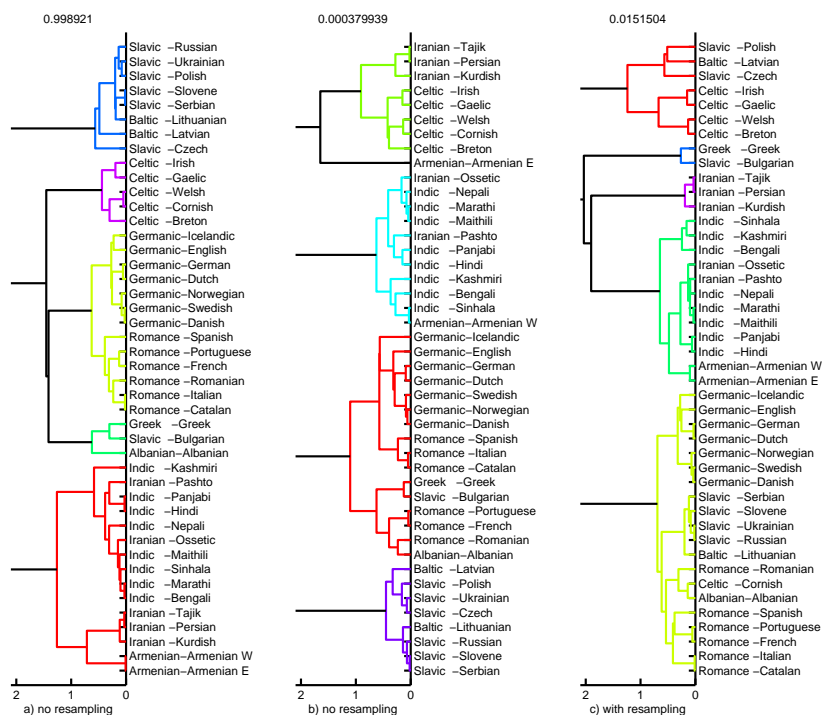

Figure 4: Tree structure infered from WALS data. **(a)**,**(b)** Samples from a run with 7750 particles without resampling. **(c)** Sample from a run with resampling. The values above the trees are normalized weights. Note that the weight of (a) is almost one, which means that the contribution from the rest of the particles is infinitesimal although the tree structure in (b) also seem to capture the similarities between languages.

[2] R. M. Neal. Defining priors for distributions using Dirichlet diffusion trees. Technical Report 0104, Department of Statistics, University of Toronto, 2001.

[3] C. K. I. Williams. A MCMC approach to hierarchical mixture modelling. In *Advances in Neural Information Processing Systems*, volume 12, 2000.

[4] J. F. C. Kingman. On the genealogy of large populations. *Journal of Applied Probability*, 19:27–43, 1982. Essays in Statistical Science.

[5] J. F. C. Kingman. The coalescent. *Stochastic Processes and their Applications*, 13:235–248, 1982.

[6] J. Felsenstein. Evolutionary trees from DNA sequences: a maximum likelihood approach. *Journal of Molecular Evolution*, 17:368–376, 1981.

[7] R. C. Griffiths and S. Tavare. Simulating probability distributions in the coalescent. *Theoretical Population Biology*, 46:131–159, 1994.

[8] M. Stephens and P. Donnelly. Inference in molecular population genetics. *Journal of the Royal Statistical Society*, 62:605–655, 2000.

[9] W.R. Gilks and P. Wild. Adaptive rejection sampling for Gibbs sampling. *Applied Statistics*, 41:337–348, 1992.

[10] D. Görür and Y.W. Teh. Concave convex adaptive rejection sampling. Technical report, Gatsby Computational Neuroscience Unit, 2008.

[11] Y. Chen, J. Xie, and J. Liu. Stopping-time resampling for sequential monte carlo methods. *Journal of the Royal Statistical Society*, 67, 2005.

[12] P. Fearnhead. *Sequential Monte Carlo Method in Filter Theory*. PhD thesis, Merton College, University of Oxford, 1998.

